# Tractable Objectives for Robust Policy Optimization

**Katherine Chen**
University of Alberta
kchen4@ualberta.ca

**Michael Bowling**
University of Alberta
bowling@cs.ualberta.ca

## Abstract

Robust policy optimization acknowledges that risk-aversion plays a vital role in real-world decision-making. When faced with uncertainty about the effects of actions, the policy that maximizes expected utility over the unknown parameters of the system may also carry with it a risk of intolerably poor performance. One might prefer to accept lower utility in expectation in order to avoid, or reduce the likelihood of, unacceptable levels of utility under harmful parameter realizations. In this paper, we take a Bayesian approach to parameter uncertainty, but unlike other methods avoid making any distributional assumptions about the form of this uncertainty. Instead we focus on identifying optimization objectives for which solutions can be efficiently approximated. We introduce percentile measures: a very general class of objectives for robust policy optimization, which encompasses most existing approaches, including ones known to be intractable. We then introduce a broad subclass of this family for which robust policies can be approximated efficiently. Finally, we frame these objectives in the context of a two-player, zero-sum, extensive-form game and employ a no-regret algorithm to approximate an optimal policy, with computation only polynomial in the number of states and actions of the MDP.

## 1 Introduction

Reinforcement learning is focused on learning optimal policies from trajectories of data. One common approach is to build a Markov decision process (MDP) with parameters (i.e., rewards and transition probabilities) learned from data, and then find an optimal policy: a sequence of actions that would maximize expected cumulative reward in that MDP. However, optimal policies are sensitive to the estimated reward and transition parameters. The optimal performance on the estimated MDP is unlikely to be actually attained under the true, but unknown, parameter values. Furthermore, optimizing for the estimated parameter realization may risk unacceptable performance under other less likely parameter realizations. For example, consider a data-driven medical decision support setting: given one-step trajectory data from a controlled trial, the goal is to identify an effective treatment policy. The policy that maximizes expected utility under a single estimated model, or even averaged over a distribution of models, may still result in poor outcomes for a substantial minority of patients. What is called for is a policy that is more robust to the uncertainties of individual patients.

There are two main approaches for finding robust policies in MDPs with parameter uncertainty. The first assumes rewards and transitions belong to a known and compact uncertainty set, which also includes a single nominal parameter setting that is thought most likely to occur [19]. Robustness, in this context, is a policy's performance under worst-case parameter realizations from the set and is something one must trade-off against how well a policy performs under the nominal parameters. In many cases, the robust policies found are overly conservative because they do not take into account how likely it is for an agent to encounter worst-case parameters. The second approach takes a Bayesian perspective on parameter uncertainty, where a prior distribution over the parameter values is assumed to be given, with a goal to optimize the performance for a particular percentile [4]. Unfortunately, the approach assumes specific distributions of parameter uncertainty in order to be

tractable, e.g., rewards from Gaussians and transition probabilities from independent Dirichlets. In fact, percentile optimization with general parameter uncertainty is NP-hard [3].

In this paper we focus on the Bayesian setting where a distribution over the parameters of the MDP is given. Rather than restricting the form of the distribution in order to achieve tractable algorithms, we consider general parameter uncertainty, and instead explore the space of possible objectives. We introduce a generalization of percentile optimization with objectives defined by a measure over percentiles instead of a single percentile. This family of objectives subsumes tractable objectives such as optimizing for expected value, worst-case, or Conditional Value-at-Risk; as well as intractable objectives such as optimizing for a single specific percentile (percentile optimization or Value-at-Risk). We then introduce a particular family of percentile measures, which can be efficiently approximated. We show this by framing the problem as a two-player, zero-sum, extensive-form game, and then employing a form of counterfactual regret minimization to find near-optimal policies in time polynomial in the number of states and actions in the MDP. We give a further generalization of this family by proving a general, but sufficient, condition under which percentile measures admit efficient optimization. Finally, we empirically demonstrate our algorithm on a synthetic uncertain MDP setting inspired by finding robust policies for diabetes management.

## 2 Background

We begin with an overview of Markov decision processes and existing techniques for dealing with uncertainty in the parameters of the underlying MDP. In section 3, we show that many of the objectives described here are special cases of percentile measures.

### 2.1 Markov Decision Processes

A finite-horizon **Markov decision process** is a tuple $\mathcal{M} = \langle S, A, R, P, H \rangle$. $S$ is a finite set of states, $A$ is a finite set of actions, and $H$ is the horizon. The decision agent starts in an initial state $s_0$, drawn from an initial state distribution $P(s_0)$. System dynamics are defined by $P(s, a, s') = \mathbb{P}(s'|s, a)$ which indicates the probability of transitioning from one state $s \in S$ to another state $s' \in S$ after taking action $a \in A$. The immediate reward for being in a state and taking an action is defined by the reward function $R : S \times A \mapsto \mathbb{R}$. We will assume the rewards are bounded so that $|R(s, a)| \leq \Delta/2$. We denote $\Pi^{HR}$ as the set of all **history-dependent randomized policies**, i.e., those that map sequences of state-action pairs and the current state to probability distribution over actions. We denote $\Pi^{MR}$ as the set of all **Markov randomized policies**, i.e., those that map only the current state and timestep to a probability distribution over actions. For a fixed MDP $\mathcal{M}$, the objective is to compute a policy $\pi$ that maximizes **expected cumulative reward**,

$$V_{\mathcal{M}}^{\pi} = \mathbb{E}\left[\sum_{t=0}^{H} R(s_t, a_t) \middle| \mathcal{M}, s_0 \propto P(s_0), \pi\right] \tag{1}$$

For a fixed MDP, the set of Markov random policies (in fact, Markov deterministic policies) contains a maximizing policy. This is called the optimal policy for the fixed MDP: $\pi^* = \text{argmax}_{\pi \in \Pi^{MR}} V_{\mathcal{M}}^{\pi}$. However, for MDPs with parameter uncertainty, Markov random policies may not be a sufficient class. We will return to this issue again when discussing our own work.

### 2.2 MDPs with Parameter Uncertainty

In this paper, we are interested in the situation where the MDP parameters, $R$ and $P$, are not known. In general, we call this an uncertain MDP. The form of this uncertainty and associated optimization objectives has been the topic of a number of papers.

**Uncertainty Set Approach.** One formulation for parameter uncertainty assumes that the parameters are taken from uncertainty sets $R \in \mathcal{R}$ and $P \in \mathcal{P}$ [12]. In the robust MDP approach the desired policy maximizes performance in the worst-case parameters of the uncertainty sets:

$$\pi^* = \underset{\pi}{\text{argmax}} \min_{R \in \mathcal{R}, P \in \mathcal{P}} V_{\mathcal{M}}^{\pi} \tag{2}$$

The robust MDP objective has been criticized for being overly-conservative as it focuses entirely on the worst-case [19]. A further refinement is to assume that a nominal fixed MDP model is also given, which is thought to be a good guess for the true model. A mixed optimization objective is then proposed that trades-off between the nominal performance and robust (worst-case) performance [19]. However, neither the robust MDP objective nor the mixed objective consider a policy's performance in parameter realizations other than the nominal- and worst-cases, and neither considers the relative likelihood of encountering these parameter realizations.

Xu and Mannor [20] propose a further alternative by placing parameter realizations into nested uncertainty sets, each associated with a probability of drawing a parameter realization from the set. They then propose a distributional robustness approach, which maximizes the expected performance over the worst-case distribution of parameters that satisfies the probability bounds on uncertainty sets. This approach is a step between the specification of uncertainty sets and a Bayesian approach with a fully specified MDP parameter distribution.

**Bayesian Uncertainty Approach.** The alternative formulation to uncertainty sets, is to assume that the true parameters of the MDP, $R^*$ and $P^*$, are distributed according to a known distribution $\mathbb{P}(R, P)$. A worst-case analysis in such a formulation is non-sensical, except in the case of distributions with bounded support (i.e. Uniform distributions), in which case it offers nothing over uncertainty sets. A natural alternative is to look at percentile optimization [4]. For a fixed $\eta$, the objective is to seek a policy that will maximize the performance on $\eta$ percent of parameter realizations. Formally, this results in the following optimization:

$$\pi^* = \operatorname*{argmax}_{\pi} \max_{y \in \mathbb{R}} \quad y$$
$$\text{subject to} \quad \mathbb{P}_{\mathcal{M}}[V_{\mathcal{M}}^{\pi} \geq y] \geq \eta \tag{3}$$

The optimal policy $\pi^*$ guarantees the optimal value $y^*$ is achieved with probability $\eta$ given the distribution over parameters $\mathbb{P}(R, P)$. Delage and Mannor showed that for general reward and/or transition uncertainty, percentile optimization is NP-hard (even for a small fixed horizon) [3]. They did show that for Gaussian reward uncertainty, the optimization can be efficiently solved as a second order cone program. They also showed that for transitions with independent Dirichlet distributions that are sufficiently-peaked (e.g., given enough observations), optimizing an approximation of the expected performance over the parameters approximately optimizes for percentile performance [4].

**Objectives from Financial Economics.** Value-at-Risk (VaR) and Conditional Value-at-Risk (CVaR) are optimization objectives used to assess the risk of financial portfolios. Value-at-Risk is equivalent to percentile optimization and is intractable for general forms of parameter uncertainty. Additionally, it is not a *coherent* risk measure in that it does not follow *subadditivity*, a key coherence property that states that the risk of a combined portfolio must be no larger than the sum of the risks of its components. In contrast, Conditional Value-at-Risk at the $\eta\%$ level is defined as the "average of the $\eta \cdot 100$ worst losses"[1]. It is both a coherent and a tractable objective [13]. In section 3 we show that CVaR is also encompassed by percentile measures.

**Restrictions on Parameter Uncertainty.** One commonality among previous approaches is that they all make heavy restrictions on the form of parameter uncertainty in order to obtain efficient algorithms. A common requirement, for example, is that the uncertainty between states is uncoupled or independent; or that reward and transition uncertainty themselves are uncoupled or independent. A very recent paper relaxes this coupling in the context of uncertainty sets, however the relaxation still takes a very specific form allowing for a finite number of deviations [9]. Another common assumption is that the uncertainty is non-stationary, i.e., a state's parameter realization can vary independently with each visit. The Delage and Mannor work on percentile optimization [4] makes the more natural assumption that the uncertain parameters are stationary, but in turn requires very specific choices for the uncertainty distributions themselves. In this work, we avoid making assumptions on the form of parameter uncertainty beyond the ability to sample from the distribution. Instead, we focus on identifying the possible optimality criteria which admit efficient algorithms.

## 3 Percentile Measures

We take the Bayesian approach to uncertainty where the true MDP parameters are assumed to be distributed according to a known distribution, i.e., the true MDP $\mathcal{M}^*$ is distributed according to an

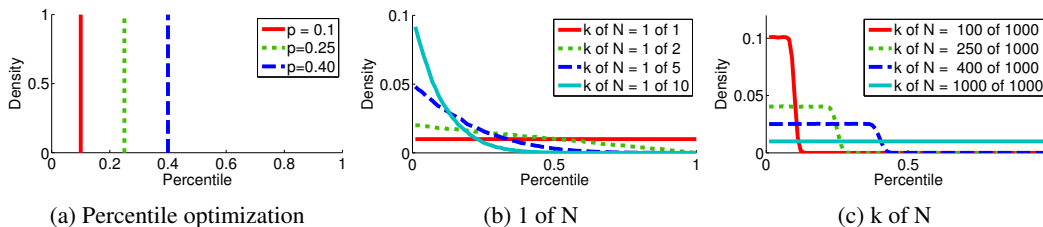

Figure 1: Examples of percentile measures.

arbitrary distribution $\mathbb{P}(\mathcal{M})$. We begin by delineating a family of objectives for robust policy optimization, which generalizes the concept of percentile optimization. While percentile optimization is already known to be NP-hard, in section 4, we will restrict our focus to a subclass of our family that does admit efficient algorithms. Rather than seeking to maximize one specific percentile of MDPs, our family of objectives maximizes an integral of a policy's performance over all percentiles $\eta \in [0,1]$ of MDPs $\mathcal{M}$ as weighted by a **percentile measure** $\mu$. Formally, given a measure $\mu$ over the interval $[0,1]$ a $\mu$-robust policy is the solution to the following optimization:

$$\pi^* = \underset{\pi \in \Pi}{\operatorname{argmax}} \sup_{y \in \mathcal{F}} \int_\eta y(\eta) d\mu$$
$$\text{subject to} \quad \mathbb{P}_{\mathcal{M}}[V_{\mathcal{M}}^\pi \geq y(\eta)] \geq \eta \quad \forall \eta \in [0,1] \tag{4}$$

where $\mathcal{F}$ is the class of real-valued, bounded, $\mu$-integrable functions on the interval $[0,1]$.

There are many possible ways to choose the measure $\mu$, each of which corresponds to a different robustness interpretation and degree. In fact, our distribution measures framework encompasses optimization objectives for the expected, robust, and percentile MDP problems as well as for VaR and CVaR. In particular, if $\mu$ is the Lebesgue measure (i.e., a uniform density over the unit interval), all percentiles are equally weighted and the $\mu$-robust policy will optimize the expected cumulative reward over the distribution $\mathbb{P}(\mathcal{M})$. In other words, it maximizes $E_{\mathcal{M}}[V_{\mathcal{M}}^\pi]$. This objective was explored by Mannor et al. [10], where they concluded that the common approach of computing an optimal policy for the expected MDP, i.e., maximizing $V_{E[\mathcal{M}]}^\pi$, results in a biased optimization of the desired value expectation under general transition uncertainty. Alternatively, when $\mu = \delta_{0.1}$, where $\delta_\eta$ is the Dirac delta at $\eta$, the optimization problem becomes identical to the VaR and percentile optimization problems where $\eta = 0.1$, the 10th percentile. The measures for the 10th, 25th, and 40th percentiles are shown in figure 1a. When $\mu = \delta_0$, the optimization problem becomes the worst-case robust MDP problem, over the support of the distribution $\mathbb{P}(\mathcal{M})$. Finally, if $\mu$ is a decreasing step function at $\eta$, this corresponds to the CVaR objective at the $\eta\%$ level, with equal weighting for the bottom $\eta$ percentiles and zero weighting elsewhere.

## 4 $k$-of-$N$ Measures

There is little reason to restrict ourselves to percentile measures that put uniform weight on all percentiles, or Dirac deltas on the worst-case or specific percentiles. One can imagine creating other density functions over percentiles, and not all of these percentile measures will necessarily be intractable like percentile optimization. In this section we introduce a subclass of percentile measures, called $k$-of-$N$ measures, and go on to show that we can efficiently approximate $\mu$-robust policies for this entire subclass.

We start by imagining a sampling scheme for evaluating the robustness of a fixed policy $\pi$. Consider sampling $N = 1000$ MDPs from the distribution $\mathbb{P}(\mathcal{M})$. For each MDP we can evaluate the policy $\pi$ and then rank the MDPs based on how much expected cumulative reward $\pi$ attains on each. If we choose to evaluate our policy based on the very worst of these MDPs, that is, the $k = 1$ of the $N = 1000$ MDPs, then we get a loose estimate of the percentile value of $\pi$ in the neighborhood of the $1/1000th$ percentile for the distribution $\mathbb{P}(\mathcal{M})$. If we sample just $N = 1$ MDP, then we get an estimate of $\pi$'s expected return over the distribution. Each choice of $N$ results in a different density, and corresponding measure, over the percentiles on the interval $[0,1]$. Figure 1b depicts the shape of the density when we hold $k = 1$ while increasing the number of MDPs we sample, $N$. We see that as

$N$ increases, the policy puts more weight on optimizing for lower percentiles of MDPs. Thus we can smoothly transition from finding policies that perform well in expectation (no robustness) to policies that care almost only about worst-case performance (overly conservative robustness). Alternatively, after sampling $N$ MDPs we could instead choose the expected cumulative reward of a random MDP from the $k \geq 1$ least-favorable MDPs for $\pi$. For every choice of $k$ and $N$, this gives a different density function and associated measure. Figure 1c shows the density function for $N = 1000$ while increasing $k$. The densities themselves act as approximate step-functions whose weight falls off in the neighborhood of the percentile $\eta = k/N$. Furthermore, as $N$ increases, the shape of the density more closely approximates a step-function, and thus more closely approximates the CVaR objective. For a particular $N$ and $k$, we call this measure the $k$-**of**-$N$ measure, or $\mu_{k\text{-of-}N}$.

**Proposition 1.** *For any $1 \leq k \leq N$, the density $g$ of the measure $\mu_{k\text{-of-}N}$ is $g(\eta) \propto 1 - I_\eta(k, N-k)$, where $I_x(\alpha, \beta) = B(x; \alpha, \beta)/B(\alpha, \beta)$ is the regularized incomplete Beta function.*

The proof can be found in the supplemental material.

### 4.1 $k$-**of**-$N$ **Game**

Our sampling description of the $k$-of-$N$ measure can be reframed as a two-player zero-sum extensive-form game with imperfect information, as shown in Figure 2. Each node in the tree represents a game state or history labeled with the player whose turn it is to act, with each branch being a possible action.

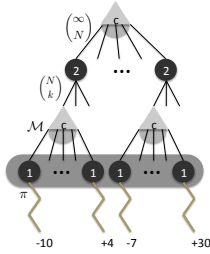

Figure 2: $k$-of-$N$ game tree

In our game formulation, chance, denoted as player $c$, first selects $N$ MDPs according to $\mathbb{P}(\mathcal{M})$. The adversary, denoted as player 2, has only one decision in the game which is to select a subset of $k$ MDPs out of the $N$, from which chance selects one MDP $\mathcal{M}$ uniformly at random. At this point, the decision maker, denoted as player 1, has no knowledge of the sampled MDPs, the choice made by the adversary, or the final selected MDP. Hence, player 1 might be in any one of the circled nodes and can not distinguish one from the other. Such histories are partitioned into one set, termed an **information set**, and the player's policy must be identical for all histories in an information set. The decision maker now alternates turns with chance, observing states sampled by chance according to the chosen MDP's transition function, but not ever observing the chosen MDP itself, i.e., histories with the same sequence of sampled states and chosen actions belong to the same information set for player 1. After the horizon has been reached, the utility of the leaf node is just the sum of the immediate rewards of the decision maker's actions according to the chosen MDP's reward function.

The decision maker's behavioral strategy in the game maps information sets of the game to a distribution over actions. Since the only information is the observed state-action sequence, the strategy can be viewed as a policy in $\Pi^{HR}$ (or possibly $\Pi^{MR}$, as we will discuss below).

Because the $k$-of-$N$ game is zero-sum, a Nash equilibrium policy in the game is one that maximizes its expected utility against its best-response adversary. The best-response adversary for any policy is the one that chooses the $k$ least favorable MDPs for that policy. Thus a policy's value against its best-response is, in fact, its value under the measure $\mu_{k\text{-of-}N}$. Hence, a Nash equilibrium policy for the $k$-of-$N$ game is a $\mu_{k\text{-of-}N}$-robust policy. Furthermore, an $\epsilon$-Nash equilibrium policy is a $2\epsilon$ approximation of a $\mu_{k\text{-of-}N}$-robust policy.

### 4.2 Solving $k$-**of**-$N$ **Games**

In the past five years there have been dramatic advances in solving large zero-sum extensive-form games with imperfect information [21, 5, 8]. These algorithmic advancements have made it possible to solve games five orders of magnitude larger than previously possible. Counterfactual regret minimization (CFR) is one such approach [21]. CFR is an efficient form of regret minimization for extensive-form games. Its use in solving extensive-form games is based on the principle that two no-regret learning algorithms in self-play will have their average strategies converge to a Nash equilibrium. However, the $k$-of-$N$ game presents a difficulty due to the imbalance in the size of the two players' strategies. While player one's strategy is tractable (the size of a policy in the underly-

ing MDP), player two's strategy involves decisions at infinitely many information sets (one for each sampled set of $N$ MDPs).

A recent variant of CFR, called CFR-BR, specifically addresses the challenge of an adversary having an intractably large strategy space [6]. It combines two ideas. First, it avoids representing the entirety of the second player's strategy space, by having the player always play according to a best-response to the first player's strategy. So, the repeated games now involve a CFR algorithm playing against its own best-response. Note that best-response is also a regret-minimizing strategy, and so such repeated play still converges to a Nash equilibrium. Second, it avoids having to compute or store a complete best-response by employing sampling over chance outcomes to focus the best-response and regret updates on a small subtree of the game on each iteration. The approach removes all dependence on the size of the adversary's strategy space in either computation time or memory. Furthermore, it can be shown that the player's current strategy is approaching almost-always a Nash equilibrium strategy, and so there is no need to worry about strategy averaging. CFR-BR has the following convergence guarantee.

**Theorem 1** (Theorems 4 and 6 [6]). *For any $p \in (0,1]$, after $T^*$ iterations of chance-sampled CFR-BR where $T^*$ is chosen uniformly at random from $\{1, \ldots, T\}$, with probability $(1-p)$, player 1's strategy on iteration $T^*$ is part of an $\epsilon$-Nash equilibrium with*

$$\epsilon = \left(1 + \frac{2}{\sqrt{p}}\right) \frac{2H\Delta |\mathcal{I}_1| \sqrt{|A_1|}}{p\sqrt{T}}$$

*where $H\Delta$ is the maximum difference in total reward over $H$ steps, and $|\mathcal{I}_1|$ is the number of information sets for player 1.*

The key property of this theorem is that the bound is decreasing with the number of iterations $T$ and there is no dependence on the size of the adversary's strategy space. The random stopping time of the algorithm is unusual and is needed for the high-probability guarantee. Johanson and colleagues note, "In practice, our stopping time is dictated by convenience and availability of computational resources, and so is expected to be sufficiently random." [6]; we follow this practice.

The application of chance-sampled CFR-BR to $k$-of-$N$ games is straightforward. The algorithm is iterative. On each iteration, $N$ MDPs are sampled from the uncertainty distribution. The best-response for this subtree of the game involves simply evaluating the player's current MDP policy on the $N$ MDPs and choosing the least-favorable $k$. Chance samples again, by choosing a single MDP from the least-favorable $k$. The player's regrets are then updated using the transitions and rewards for the selected MDP, resulting in a new policy for the next iteration. See the supplemental material for complete details.

**Markovian Policies and Imperfect Recall.** There still remains one important detail that we have not discussed: the nature and size of player 1's strategy space. In finite horizon MDPs with no parameter uncertainty, an optimal policy exists in the space of Markovian policies ($\Pi^{MR}$) — policies that depend only on the number of timesteps remaining and the current state, but not on the history of past states and actions. Under transition uncertainty, this is no longer true. The sequence of past states and actions provide information about the uncertain transition parameters, which is informative for future transitions. For this case, optimal policies are not in general Markovian policies as they will depend upon the entire history of states and actions ($\Pi^{HR}$). As a result, the number of information sets (i.e., decision points) in an optimal policy is $|\mathcal{I}_1| = |S|((|S||A|)^H - 1)/(|S||A| - 1)$, and so polynomial in the number of states and actions for any fixed horizon, but exponential in the horizon itself. While being exponential in the horizon may seem like a problem, there are many interesting real-world problems with short time horizons. One such class of problems is **Adaptive treatment strategies** (ATS) for sequential medical treatment decisions [11, 15]. Many ATS problems have time horizons of $H \leq 3$, e.g., CATIE ($H = 2$) [16, 17] and STAR*D ($H = 3$) [14].

Under reward uncertainty (where rewards are not observed by the agent while acting), the sequence of past states and actions is not informative, and so Markovian policies again suffice.[1] In this case, the number of information sets $|\mathcal{I}_1| = |S|H$, and so polynomial in both states and the horizon. However, such an information-set structure for the player results in a game with **imperfect recall**,

where the player forgets information (past states and actions) it previously knew. Perfect recall is a fundamental requirement for extensive-form game solvers. However, a recent result has presented sufficient conditions under which the perfect recall assumption can be relaxed and CFR will still minimize overall regret [7]. These conditions are exactly satisfied in the case of reward uncertainty: the forgotten information (i) does not influence future rewards, (ii) does not influence future transition probabilities, (iii) is never known by the opponent, (iv) is not remembered later by the player. Therefore, we can construct the extensive-form game with the player restricted to Markovian policies and still solve it with CFR-BR.

**CFR-BR for $k$-of-$N$ Games.** We can now analyze the use of CFR-BR for computing approximate $\mu_{k\text{-of-}N}$-robust policies.

**Theorem 2.** *For any $\epsilon > 0$ and $p \in (0, 1]$, let,*

$$T = \left(1 + \frac{2}{\sqrt{p}}\right)^2 \frac{16H^2\Delta^2|\mathcal{I}_1|^2|A|}{p^2\epsilon^2}.$$

*With probability $1 - p$, when applying CFR-BR to the $k$-of-$N$ game, its current strategy at iteration $T^*$, chosen uniformly at random in the interval $[1, T]$, is an $\epsilon$-approximation to a $\mu_{k\text{-of-}N}$-robust policy. The total time complexity is $O\left((H\Delta/\epsilon)^2 \frac{|\mathcal{I}_1|^3|A|^3 N \log N}{p^3}\right)$, where $|\mathcal{I}_1| \in O(|S|H)$ for arbitrary reward uncertainty and $|\mathcal{I}_1| \in O(|S|^{H+1}|A|^H)$ for arbitrary transition and reward uncertainty.*

*Proof.* The proof follows almost directly from Theorem 1 and our connection between $k$-of-$N$ games and the $\mu_{k\text{-of-}N}$ measure. The choice of $T$ by Theorem 1 guarantees the policy is an $\epsilon/2$-Nash approximation, which in turn guarantees the policy is within $\epsilon$ of optimal in the worst-case, and so is an $\epsilon$ approximation to a $\mu_{k\text{-of-}N}$-robust policy. Each iteration requires $N$ policy evaluations each requiring $O(|\mathcal{I}_1||A|)$ time; these are then sorted in $O(N \log N)$ time; and finally the regret update in $O(|\mathcal{I}_1||A|)$ time. Theorem 2 gives us our overall time bound. $\square$

## 5 Non-Increasing Measures

We have defined a family of percentile measures, $\mu_{k\text{-of-}N}$, that represent optimization objectives that differ in how much weight they place on different percentiles and can be solved efficiently. In this section, we go beyond our family of measures and provide a very broad but still sufficient condition for which a measure can be solved efficiently. We conjecture that a form of this condition is also necessary, but leave that for future work.

**Theorem 3.** *Let $\mu$ be an absolutely continuous measure with density function $g_\mu$, such that $g_\mu$ is non-increasing and piecewise Lipschitz continuous with $m$ pieces and Lipschitz constant $L$. A $\mu$-robust policy can be approximated with high probability in time polynomial in $\{|A|, |S|, \Delta, L, m, \frac{1}{\epsilon}, \frac{1}{p}\}$ for (i) arbitrary reward uncertainty with time also polynomial in the horizon or (ii) arbitrary transition and reward uncertainty with a fixed horizon.*

The proof is in the supplemental material. Note that previously known measures with efficient solutions (i.e., worst-case, expectation-maximization, and CVaR) satisfy the property that the weight placed on a particular percentile is never smaller than a larger percentile. Our $k$-of-$N$ measures also have this property. Percentile measures ($\eta > 0$), though, do not: they place infinitely more weight on the $p$ percentile than any of the percentiles less than $\eta$. At the very least, we have captured the condition that separates the currently known-to-be-easy measures from the currently known-to-be-hard ones.

## 6 Experiments

We now explore our $k$-of-$N$ approach in a simplified version of a diabetes management task. Our results aim to demonstrate two things: first, that CFR-BR can find $k$-of-$N$ policies for MDP problems with general uncertainty in rewards and transitions; and second, that optimizing for different percentile measures creates policies that differ accordingly.

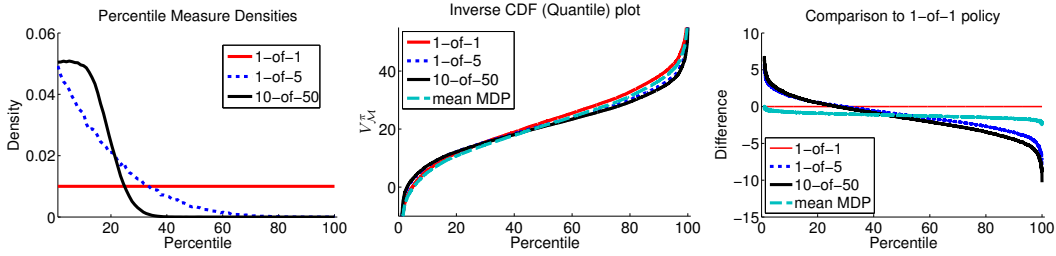

Figure 3: Evaluation of $k$-of-$N$ percentile measures on the diabetes management task.

Our simplified diabetes management MDP simulates the daily life of a diabetic patient distilled into a small MDP with $|S| = 9$ states, $|A| = 3$ actions and a time horizon of $H = 3$. States are a combination of blood glucose level and meal size. Three times daily, corresponding to meal times, the patient injects themselves with a dose of insulin to bring down the rise in blood glucose that comes with consuming carbohydrates at each meal. A good treatment policy keeps blood glucose in the moderate range all day. The uncertain reward function is sampled from a independent multivariate Normal distribution and transition probabilities are sampled from Dirichlet distributions, but both could have been drawn from other distributions. The Dirichlet parameter vector is the product of a fixed set of per-state parameters with an MDP-wide multiplicative factor $q \sim \text{Unif}[1, 5]$ to simulate variation in patient sensitivity to insulin, and results in transition uncertainty between states that is not independent. For full details on the problem set up, see the supplemental material.

We used CFR-BR to find optimal policies for the 1-of-1, 1-of-5, and 10-of-50 percentile measures. The densities for these measures are shown in Figure 3(left). We also computed the policy that optimizes $V^{\pi}_{\mathbb{E}(\mathcal{M})}$, that is the optimal policy for the **mean MDP**. We evaluated the performance of all of these policies empirically on over 10,000 sampled MDPs and show the empirical quantile function (inverse CFR) in Figure 3(center). To highlight the differences between these policies, we show the performance of the policies relative to the 1-of-1-robust policy over the full range of percentiles in Figure 3(right). From the difference plot, we see that the optimal policy for the mean MDP, although optimal for the mean MDP's specific parameters, does not perform well over the uncertainty distribution (as noted in [10]). All of the $k$-of-$N$ policies are more robust, performing better on the lower percentiles, while 1-of-1 is almost a uniform improvement. We also see that 1-of-5 and 10-of-50 policies perform quite differently despite having the same $k/N$ ratio. Because the 10-of-50 policy has a sharper drop-off in density at the 20th percentile compared to the 1-of-5 policy, we see that 10-of-50 policies give up more performance in higher percentile MDPs for a bit more performance in the lowest 20 percentile MDPs compared to the 1-of-5 policy.

# 7   Conclusion

This is the first work we are aware of to do robust policy optimization with general parameter uncertainty. We describe a broad family of robustness objectives that can be efficiently optimized, and present an algorithm based on techniques for Nash approximation in imperfect information extensive-form games. We believe this approach will be useful for adaptive treatment strategy optimization, where small sample sizes cause real parameter uncertainty and the short time horizons make even transition uncertainty tractable. The next step in this direction is to extend these robustness techniques to large, or continuous state-action spaces. Abstraction has proven useful for finding good policies in other large extensive-form-games [2, 18], and so will likely prove effective here.

# 8   Acknowledgements

We would like to thank Kevin Waugh, Anna Koop, the Computer Poker Research Group at the University of Alberta, and the anonymous reviewers for their helpful discussions. This research was supported by NSERC, Alberta Innovates Technology Futures, and the use of computing resources provided by WestGrid and Compute/Calcul Canada.

## Footnotes

[1]Markovian policies are also sufficient under a non-stationary uncertainty model, where the transition parameters are resampled independently on repeated visits to states (see the end of Section 2.2).

# References

[1] Carlo Acerbi. Spectral Measures of Risk: a Coherent Representation of Subjective Risk Aversion. *Journal of Baking and Finance*, 2002.

[2] D. Billings, N. Burch, A. Davidson, R. Holte, J. Schaeffer, T. Schauenberg, and D. Szafron. Approximating game-theoretic optimal strategies for full-scale poker. *Proceedings of the Eighteenth International Joint Conference on Artificial Intelligence (IJCAI)*, 2003.

[3] Erick Delage. *Distributionally Robust Optimization in context of Data-driven Problems*. PhD thesis, Stanford University, 2009.

[4] Erick Delage and Shie Mannor. Percentile Optimization in Uncertain Markov decision processes with Application to Efficient Exploration. *Proceedings of the 24th International Conference on Machine Learning (ICML)*, 2007.

[5] Samid Hoda, Andrew Gilpin, Javier Peña, and Tuomas Sandholm. Smoothing techniques for computing Nash equilibria of sequential games. *Mathematics of Operations Research*, 35(2):494–512, 2010.

[6] Michael Johanson, Nolan Bard, Neil Burch, and Michael Bowling. Finding optimal abstract strategies in extensive-form games. *Proceedings of the 26th Conference on Artificial Intelligence (AAAI)*, 2012.

[7] Marc Lanctot, Richard Gibson, Neil Burch, Martin Zinkevich, and Michael Bowling. No-regret learning in extensive-form games with imperfect recall. *Proceedings of the 29th International Conference on Machine Learning (ICML)*, 2012.

[8] Marc Lanctot, Kevin Waugh, Martin Zinkevich, and Michael Bowling. Monte Carlo Sampling for Regret Minimization in Extensive Games. *Advances in Neural Information Processing Systems (NIPS)*, 2009.

[9] Shie Mannor, Ofir Mebel, and Huan Xu. Lighting Does Not Strike Twice: Robust MDPs with Coupled Uncertainty. *Proceedings of the 29th International Conference on Machine Learning (ICML)*, 2012.

[10] Shie Mannor, Duncan Simester, Peng Sun, and John N. Tsitsiklis. Bias and variance in value function estimation. *Management Science*, 53(2):308–322, February 2007.

[11] Susan A. Murphy and James R. McKay. Adaptive treatment strategies: an emerging approach for improving treatment effectiveness. *(Newsletter of the American Psychological Association Division 12, section III: The Society for the Science of Clinical Psychology)*, 2003.

[12] Arnab Nilim and Laurent El Ghaoui. Robust Control of Markov decision processes with Uncertain Transition matrices. *Operations Research*, 53(5):780–798, October 2006.

[13] R. Tyrrell Rockafellar and Stanislav Uryasev. Conditional value-at-risk for general loss distributions. *Journal of Baking and Finance*, 26:1443–1471, 2002.

[14] A. J. Rush, M. Fava, S. R. Wisniewski, P.W. Lavori, M. H. Trivedi, H. A. Sackeim, M. E. Thase, A. A. Nierenberg, F. M. Quitkin, T.M. Kashner, D.J. Kupfer, J. F. Rosenbaum, J. Alpert, J. W. Stewart, P. J. Mc-Grath, M. M. Biggs, K. Shores-Wilson, B. D. Lebowitz, L. Ritz, and G. Niederehe. Sequenced treatment alternatives to relieve depression (STAR*D): rational and design. *Controlled Clinical Trials*, 25(1):119–142, 2004.

[15] Susan A. Shortreed, Eric Laber, Daniel J. Lizotte, T. Scott Stroup, Joelle Pineau, and Susan A. Murphy. Informing sequential clinical decision-making through Reinforcement learning: an empirical study. *Machine Learning*, 84(1-2):109–136, July 2011.

[16] T. Scott Stroup, J.P. McEvoy, M.S. Swartz, M.J. Byerly, I.D. Glick, J.M Canive, M. McGee, G.M. Simpson, M.D. Stevens, and J.A. Lieberman. The National Institute of Mental Health clinical antipschotic trials of intervention effectiveness (CATIE) project: schizophrenia trial design and protocol development. *Schizophrenia Bulletin*, 29(1):15–31, 2003.

[17] M.S. Swartz, D.O. Perkins, T.S. Stroup, J.P. McEvoy, J.M. Nieri, and D.D. Haal. Assessing clinical and functional outcomes in the clinical antipsychotic of intervention effectiveness (CATIE) schizophrenia trial. *Schizophrenia Bulletin*, 1(33-43), 29.

[18] Kevin Waugh, Martin Zinkevich, Michael Johanson, Morgan Kan, David Schnizlein, and Michael Bowling. A practical use of imperfect recall. *Proceedings of the Eighth Symposium on Abstraction, Reformulation and Approximation (SARA)*, 2009.

[19] Huan Xu and Shie Mannor. On robustness/performance tradeoffs in linear programming and markov decision processes. *Operations Research*, 2005.

[20] Huan Xu and Shie Mannor. Distributionally Robust Markov decision processes. *Advances in Neural Information Processing Systems (NIPS)*, 2010.

[21] Martin Zinkevich, Michael Johanson, Michael Bowling, and Carmelo Piccione. Regret Minimization in Games with Incomplete Information. *Advances in Neural Information Processing Systems (NIPS)*, 2008.

